# A new model of spatial representations in multimodal brain areas.

**Sophie Deneve**
Department of Brain and cognitive Science
University of Rochester
Rochester, NY 14620.
*sdeneve@bcs.rochester.edu*

**Jean-Rene Duhamel**
Institut des Sciences Cognitives
C.N.R.S
Bron, France 69675
*jrd@isc.cnrs.fr*

**Alexandre Pouget**
Department of Brain and Cognitive
University of Rochester
Rochester, NY 14620.
*alex@bcs.rochester.edu*

## Abstract

Most models of spatial representations in the cortex assume cells with limited receptive fields that are defined in a particular egocentric frame of reference. However, cells outside of primary sensory cortex are either gain modulated by postural input or partially shifting. We show that solving classical spatial tasks, like sensory prediction, multi-sensory integration, sensory-motor transformation and motor control requires more complicated intermediate representations that are not invariant in one frame of reference. We present an iterative basis function map that performs these spatial tasks optimally with gain modulated and partially shifting units, and tests it against neurophysiological and neuropsychological data.

In order to perform an action directed toward an object, it is necessary to have a representation of its spatial location. The brain must be able to use spatial cues coming from different modalities (e.g. vision, audition, touch, proprioception), combine them to infer the position of the object, and compute the appropriate movement.

These cues are in different frames of reference corresponding to different sensory or motor modalities. Visual inputs are primarily encoded in retinotopic maps, auditory inputs are encoded in head centered maps and tactile cues are encoded in skin-centered maps. Going from one frame of reference to the other might seem easy. For example, the head-centered position of an object can be approximated by the sum of its retinotopic position and the eye position. However, positions are represented by population codes in the brain, and computing a head-centered map from a retinotopic map is a more complex computation than the underlying sum. Moreover, as we get closer to sensory-motor areas it seems reasonable to assume

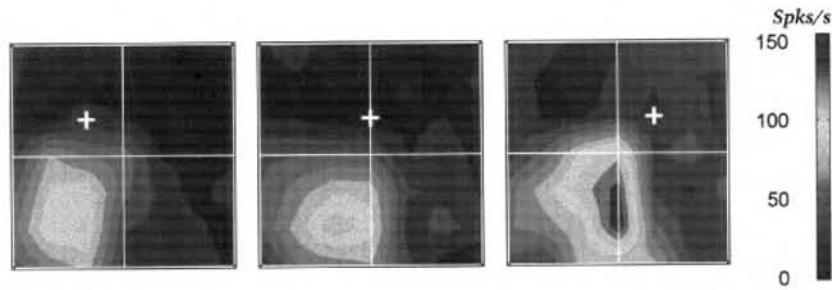

Figure 1: Response of a VIP cell to visual stimuli appearing in different part of the screen, for three different eye positions. The level of grey represent the frequency of discharge (In spikes per seconds). The white cross is the fixation point (the head is fixed). The cell's receptive field is moving with the eyes, but only partially. Here the receptive field shift is 60% of the total gaze shift. Moreover this cell is gain modulated by eye position (adapted from Duhamel et al).

that the representations should be useful for sensory-motor transformations, rather than encode an "invariant" representation.

According to the linear model, space is always represented in the sensory and sensory-motor cortex in one particular egocentric frame of reference. This process is mediated by cells whose receptive fields are anchored to a particular body part. In this view spatial cues coming from different modalities should all be remapped in a common frame of reference at some point, that can be used in turn to compute motor maps (for reaching, grasping, etc ).

The linear model was challenged when cells truly invariant in one modality failed to be found in parietal areas. Andersen et al, for example, found retinotopic cells that were gain modulated by eye position in LIP [1], but none of these cells had a head-centered receptive fields. Subsequent studies confirmed that gain-modulation by eye position is a very general phenomena in the cortex, whereas truly head-centered or arm-centered cells have rarely been reported.

More recently, in VIP, Duhamel et al. found cells that were neither eye nor head-centered, but whose receptive fields were partially moving with the eyes [2]. As a consequence, the receptive fields appeared to be moving both in the retinotopic and head-centered frames of reference (see figure 1). The amount of shift with gaze varied from cell to cell, and was continuously distributed between 0% (head-centered) and 100% (retinotopic). Partially shifting cells where also found for auditory targets in LIP [5] and in the superior colliculus [3].

We will show in this paper that the nature of the problem of integrating postural and sensory inputs from different modalities, and providing motor outputs with distributed population codes lead us to postulate the existence of these gain modulated and/or partially moving receptive fields in the associative brain areas, instead of invariant representations. We present an interconnected network that can perform multi-directional coordinate and sensory-motor transforms by using intermediate basis function units. These intermediate units are gain modulated by eye position, have partially shifting receptive field and, as a result, represent space in a mixture of frames of reference. They provide a new model of spatial representations in multimodal areas according to which cells responses are not determined solely by the position of the stimulus in a particular egocentric frame of reference, but by the

interactions between the dominant input modalities.

# 1 Sensory predictions and sensory-motor transformations with distributed population codes

We will focus on the eye/head system which deals with two frames of reference (retinotopic and head-centered) and one postural input (the eye position). Sensory predictions consist of anticipating a stimulus in one sensory modality from a stimulus originating from the same location, but in another sensory modality. Predictions of auditory stimuli from visual stimuli, for example, requires the computation of a head-centered map from a retinotopic map.

## 1.1 Coordinate transforms and sensory predictions

We assume that the tuned response of a retinotopic cell can be modeled by a Gaussian $B^r(R - R_i)$ of the distance between the stimulus position $R$ and the receptive field center $R_i$, and that the response of a postural cell to eye position can be modeled by a gaussian $B^e(E - E_j)$ of the difference between the eye position $E$ and the preferred angle $E_j$. In addition we suppose that cells are organized topographically in each layer, so that a stimulus at position $r$ and for eye position $g$ will give rise to a hill of activity peaking at position $r$ on the retinotopic map and $g$ on the eye position map. We wish to compute a head-centered map where cells responses are described by head-centered gaussian tuning curves $B^h(H - H_k)$ where H is the head-centered position and $H_k$ the preferred position.

Given the geometry of the eye/head system, we have approximately $H = R + E$, but this does not simplify the computation of coordinate transform with population codes. We certainly cannot have $B^h(H - H_k) = B^e(E - E_j) + B^r(R - R_k)$.

## 1.2 Basis function map

To solve this problem we could use an intermediate neural layer that implements a product between visual and postural tuning curves [4]. Products of Gaussians are basis functions and thus a population of retinotopic cells gain modulated by eye position, whose responses are described by $B^r(R - R_i)B^e(E - E_j)$ implement a basis function map of $R$ and $E$. Any function $f(R, E)$ can be approximated by a linear combination of these cells responses:

$$f(R, E) = \sum_{ij} w_{ij} B^r(R - R_i) B^e(E - E_j). \tag{1}$$

In particular, a head centered map is a function of retinotopic position and eye position and can be computed very easily from the basis function map (by a simple linear combination). Even more importantly, any sensory-motor transform can be implemented by feedforward weights coming from the basis function layer. The basis function map itself can be readily implemented from a retinotopic map and an eye position map, by connecting each unit with one visual cell and one eye position cell, and computing a product between these two inputs [4].

Similarly, another basis function map could be implemented by making the product between auditory and postural tuning curves, $B^r(R - R_i)B^h(H - H_k)$, in order to predict the position of a visual cue from the sound it makes, or to compute reaching toward auditory cues. However it would be better to combine these two

basis function maps in a common architecture, especially if we want to integrate visual and auditory inputs or implement motor feedback to sensory representation, both of which require a multi-directional computation.

## 2 Multi-directional coordinate transforms with distributed population codes

If we want to combine these two basis function maps without giving the priority to one modality, we can intuitively use basis functions that are a product between the three tuning curves:

$$B_{ij}(R, E, H) = B^r(R - R_i)B^e(E - E_j)B^h(H - H_{i+j}) \qquad (2)$$

From this intermediate representation, the three sensory maps $B^r(R - R_i)$, $B^e(E - E_j)$ and $B^h(H - H_{i+j})$ can be computed by simple projections. This ensures that this basis function units can use the two sensory maps as both input and output.

We implemented this idea in an interconnected neural network that non-linearly combines visual, auditory and postural inputs in an intermediate layer (the basis function map), which in turn is used to reconstruct the activities on the auditory, visual, and postural layers. This network is completely symmetric, similarly processing visual, postural and auditory inputs. It converges to stable hills of activity on the three neural maps that simultaneously gives the retinotopic position, head-centered position, and the eye position in the input (see figure 2A), performing multi-directional sensory prediction. For this reason, we called this model an iterative basis function network.

## 3 The iterative basis function network

The network is represented on figure 2A. It has four layers: three visible, one dimensional layers (visual, auditory and postural) and a two dimensional hidden layer. The three input layers are not directly connected to one another, but they are all interconnected with the hidden layer. These interconnections are symmetric, i.e. the connection between neuron A and B has the same strength as the connection between neuron B and A. This ensures that the network will converge towards a stable state.

We note $\mathbf{W}^r$, $\mathbf{W}^h$, $\mathbf{W}^e$ the respective weights of the retinotopic, head-centered and eye position layers with the hidden layer. All three weight matrices are circular symmetric gaussian filters. The connection between the $i^{th}$ unit in each input layers, and the hidden layer $l, m$ are $\mathbf{W}^r(i, l, m) = B(l - i)$, $\mathbf{W}^e(i, l, m) = B(m - i)$, $\mathbf{W}^h(i, l, m) = B((l + m) - i)$, where B is a circular gaussian matrix:

$$B(x) = Z \exp(\sigma_w(\cos(\frac{2\pi x}{N}) - 1)) \qquad (3)$$

$\sigma_w$ governs the width of the weight, $Z$ is a constant that controls the dominance of the corresponding sensory or postural modality on the intermediate layer, and N is the number of units in the input layers. Note that with these weights, the hidden unit $l, m$ is maximally connected to the unit $l$ in the retinotopic layer, $m$ in the eye position layer, and $l + m$ in the head-centered layer. This connectivity is responsible

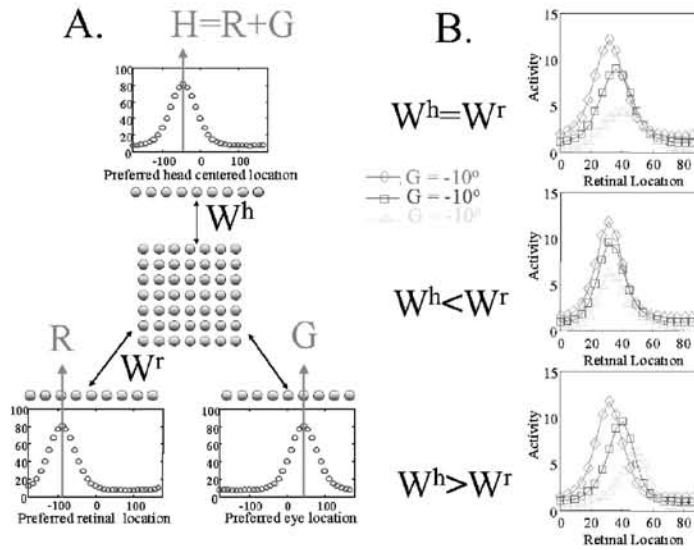

Figure 2: A- Architecture of the iterative BF map. The intermediate cells look like partially shifting cell in VIP. B- An intermediate cell's response properties when one varies the ratio $Z_r/Z_h$ of modality dominance (strength of the weights). The gain of the shift varies from 0 to 1 depending of the relative strength of $W^h$ (the auditory weights) and $W^r$ (the visual weights).

for the fact that the network will compute $H = R + E$. This approach can generalize to arbitrary mapping $M = f(R, E)$ if we replace $\mathbf{W}^h(i, l, m) = B((l + m) - i)$ by $\mathbf{W}^h(i, l, m) = B(f(l, m) - i)$.

Activities on the inputs layers are pooled linearly on the intermediate layers, according to the connection matrices. Then these pooled inputs are squared and normalized with a divisive inhibition. The resulting activities on the intermediate layer are then sent back to the input layers, through the symmetric connections, and in turn squared and normalized.

The inputs are modeled by bell-shaped distribution of activities clamped on the input layers at time 0. The amplitude of these initial hills of activity represents the contrasts of the stimuli. A purely visual stimulus, for example, would have an auditory contrast of 0 on the head-centered layer. Except for very low contrasts in all modalities, the network converges toward non-zero stable states when provided with visual, auditory, or bimodal input. These stable states are stable hills of activity on the visual, auditory and postural layers, so that the position of the hill on the head-centered layer is the sum of the position of the hill on the visual layer, and the position of the hill on the postural layer.

When provided with visual and postural input, the network predicts the auditory position of the stimulus. When provided with auditory and postural input, the retinotopic position can be read from the position of the stable hill on the visual layer. Thus, the network is automatically doing coordinate transforms in both directions. The whole process takes no more than 2 iterations.

# 4 Spatial representation in the intermediate layer

The cells in the intermediate layer provide a multimodal representation of space that we can characterize and compare to neurophysiological data. We will focus on the unit's response after the network reached its stable state. The final state depends only on the position encoded in the input, which implies that the unit's responses are identical regardless of the input modality (visual, auditory or bimodal). The receptive fields in different modalities are spatially congruent, like the receptive fields of most multimodal cells in the brain.

In figure 2B, we plotted for different eye positions the activity of an intermediate cell as a function of the retinotopic position of the stimulus. Note that because of the symmetries in the network, all the other intermediate cells responses are translated version of this one. The critical parameter that will govern the intermediate representation is ratio $Z_r/Z_h$ that defines the relative strength of visual and auditory weights. This is the only parameter we manipulated in this study.

When neither the visual nor the auditory representation dominates (that is, when $Z_r/Z_h = 1$, see figure 2B, top panel), the intermediate cell's receptive field on the retina shift with the eyes, but it does not shift as much as the eyes do. This is a partially shifting cell, gain modulated by eye position. The amount of receptive field shift with the gaze is 50%. In fact we found that this cell's response was very close to a product between a gaussian of retinotopic position, head-centered position and eye position, thus implementing the basis function we already proposed as a solution to the multi-directional computation problem. This cell looks very much like a one dimensional version of the particular VIP cell plotted in figure 1A.

Varying the ratio $x = \frac{Z_r}{Z_h}$ does not affect the performance of the network for coordinate transform (the only change occurring on the input layers is a change in the amplitude of the stable hills) but it changes the intermediate representation, particularly the amount of receptive field shift with gaze. There is a continuum between a gain modulated retinotopic cell for a high value of $x$ ( 0% shift, figure 2B, middle panel) and a gain modulated head-centered cell for a low value of $x$ (100% shift, figure 2B, bottom panel). This behavior is easy to understand: an intermediate cell receives tuned retinotopic, head-centered and eye position inputs. This three tuned inputs will more or less influence the unit's response, depending on their strength.

Thus, the whole distribution of shifts found in VIP could belong to an iterative basis function map with varying ratio between visual and "head-centered" weights. In the case of VIP, "head-centered" would correspond to tactile, as VIP is a visuo/tactile area. On the other hand, if one modality dominates in all cells (e.g. in LIP for vision), we can predict that the distribution of responses will be displaced toward the frame of reference of this modality.

# 5 Lesion of the iterative basis function map

In order to link the intermediate representation with spatial representations in the human parietal cortex, we studied the consequences of a lesion to this network.

Unilateral right parietal lesions result in a syndrome called hemineglect: The patient is slower to react to, and has difficulty detecting stimuli in the contralesional space. This is usually coupled with extinction of leftward stimuli by rightward stimuli. Two striking characteristics of hemineglect are that it is usually in a mixture of frames of reference, challenging the view that parietal cortex is a mosaic of areas devoted to spatial processing in different frames of reference. Additionally, extinction is

frequently cross-modal. For example, tactile stimuli can be extinguished by visual stimuli, suggesting that the lesioned spatial representation are themselves multi-modal.

We modeled a right parietal lesion by implementing a gradient of units in the intermediate layer, so that there are more cells tuned to contralateral retinotopic (visual) and contralateral head-centered (auditory) positions. This correspond to the observed hemispheric asymmetries in the monkey's brain.

This modification did not strongly affect the final estimates of position by the network, but the processing was slower (taking more time to reach the stable state) and the contrast threshold (minimal visual and auditory contrasts that drives the network) was higher for the leftward retinal and head-centered locations. Thus the network "neglected" stimuli in a mixture of frames of reference: The severity of neglect gradually increased from right to left both in retinotopic and head-centered coordinates.

Furthermore when we entered two simultaneous inputs to the network, we observed that the leftward stimulus was always extinguished by the rightward stimulus (the final stable state reflected only the rightward stimulus), regardless of the modality. Thus we obtained extinction of auditory stimuli by visual stimuli, and vice-versa. In our model, these two aspects of neglect (mixture of frames of reference and cross modal extinction) can be explained by a lesion in only one multimodal brain area.

## 6   Conclusion

Our approach can be easily generalized to sensory-motor transformations. In this case, the implementation of motor control (the feedback from the motor representations to the sensory representations) will lead to intermediate cells that partially shift in the sensory as well as the motor frame of reference.

This model has other (related) interesting properties that we develop elsewhere. In the presence of noisy input, it can perform optimal multi-sensory cue integration, and allows an adaptive bayesian approach to cue integration, in a biologically realistic way. Iterative basis function maps provide a new model of spatial representations and processing that can be applied to neurophysiological and neuropsychological data.

## References

[1] R. Andersen, R. Bracewell, S. Barash, J. Gnadt, and L. Fogassi. Eye position effect on visual memory and saccade-related activity in areas LIP and 7a of macaque. *Journal of Neuroscience*, 10:1176–1196, 1990.

[2] J. Duhamel, F. Bremmer, S. BenHamed, and W. Graf. Spacial invariance of visual receptive fields in parietal cortex. *Nature*, 389(6653):845–848, 1997.

[3] M. Jay and D. Sparks. Sensorimotor integration in the primate superior colliculus:1. motor convergence. *Journal of Neurophysiology*, 57:22–34, 1987.

[4] A. Pouget and T. Sejnowski. Spatial transformations in the parietal cortex using basis functions. *Journal of Cognitive Neuroscience*, 9(2), 1997.

[5] B. Stricanne, P. Mazzoni, and R. Andersen. Modulation by the eye position of auditory responses of macaque area LIP in an auditory memory saccade task. In *Society For Neuroscience Abstracts*, page 26, Washington, D.C., 1993.
